# Accelerating Bayesian Inference over Nonlinear Differential Equations with Gaussian Processes

**Ben Calderhead**
Dept. of Computing Sci.
University of Glasgow
bc@dcs.gla.ac.uk

**Mark Girolami**
Dept. of Computing Sci.
University of Glasgow
girolami@dcs.gla.ac.uk

**Neil D. Lawrence**
School of Computer Sci.
University of Manchester
neill@cs.man.ac.uk

## Abstract

Identification and comparison of nonlinear dynamical system models using noisy and sparse experimental data is a vital task in many fields, however current methods are computationally expensive and prone to error due in part to the nonlinear nature of the likelihood surfaces induced. We present an accelerated sampling procedure which enables Bayesian inference of parameters in nonlinear ordinary and delay differential equations via the novel use of Gaussian processes (GP). Our method involves GP regression over time-series data, and the resulting derivative and time delay estimates make parameter inference possible *without* solving the dynamical system explicitly, resulting in dramatic savings of computational time. We demonstrate the speed and statistical accuracy of our approach using examples of both ordinary and delay differential equations, and provide a comprehensive comparison with current state of the art methods.

## 1 Introduction

Mechanistic system modeling employing nonlinear ordinary or delay differential equations [1] (ODEs or DDEs) is oftentimes hampered by incomplete knowledge of the system structure or the specific parameter values defining the observed dynamics [16]. Bayesian, and indeed non-Bayesian, approaches for parameter estimation and model comparison [19] involve evaluating likelihood functions, which requires the explicit numerical solution of the differential equations describing the model. The computational cost of obtaining the required numerical solutions of the ODEs or DDEs can result in extremely slow running times. In this paper we present a method for performing Bayesian inference over mechanistic models by the novel use of Gaussian processes (GP) to predict the state variables of the model as well as their derivatives, thus avoiding the need to solve the system explicitly. This results in dramatically improved computational efficiency (up to four hundred times faster in the case of DDEs). We note that state space models offer an alternative approach for performing parameter inference over dynamical models particularly for on-line analysis of data, see [2]. Related to the work we present, we also note that in [6] the use of GPs has been proposed in obtaining the solution of fully parameterised linear operator equations such as ODEs. Likewise in [12] GPs are employed as emulators of the posterior response to parameter values as a means of improving the computational efficiency of a hybrid Monte Carlo sampler.

Our approach is different and builds significantly upon previous work which has investigated the use of derivative estimates to directly approximate system parameters for models described by ODEs. A spline-based approach was first suggested in [18] for smoothing experimental data and obtaining derivative estimates, which could then be used to compute a measure of mismatch for derivative values obtained from the system of equations. More recent developments of this method are described in [11]. All of these approaches, however, are plagued by similar problems. The methods

are all critically dependent on additional regularisation parameters to determine the level of data smoothing. They all exhibit the problem of providing sub-optimal point estimates; even [11] may not converge to a reasonable solution depending on the initial values selected, as we demonstrate in Section 5.1. Furthermore, it is not at all obvious how these methods can be extended for partially observed systems, which are typical in, e.g. systems biology [10, 1, 8, 19]. Finally, these methods only provide point estimates of the "correct" parameters and are unable to cope with multiple solutions. (Although it should be noted that [11] does offer a local estimate of uncertainty based on second derivatives, at additional computational cost.) It is therefore unclear how objective model comparison could be implemented using these methods.

In contrast we provide a Bayesian solution, which is capable of sampling from multimodal distributions. We demonstrate its speed and statistical accuracy and provide comparisons with the current best methods. It should also be noted that the papers mentioned above have focussed only on parameter estimation for fully observed systems of ODEs; we additionally show how parameter inference over both fully and partially observed ODE systems as well as DDEs may be performed efficiently using our state derivative approach.

## 2 Posterior Sampling by Explicit Integration of Differential Equations

A dynamical system may be described by a collection of $N$ ordinary differential equations and model parameters $\boldsymbol{\theta}$ which define a functional relationship between the process state, $\mathbf{x}(\mathbf{t})$, and its time derivative such that $\dot{\mathbf{x}}(\mathbf{t}) = \mathbf{f}(\mathbf{x}, \boldsymbol{\theta}, \mathbf{t})$. Likewise delay differential equations can be used to describe certain dynamic systems, where now an explicit time-delay $\tau$ is employed. A sequence of process observations, $\mathbf{y}(\mathbf{t})$, are usually contaminated with some measurement error which is modeled as $\mathbf{y}(\mathbf{t}) = \mathbf{x}(\mathbf{t}) + \boldsymbol{\epsilon}(\mathbf{t})$ where $\boldsymbol{\epsilon}(\mathbf{t})$ defines an appropriate multivariate noise process, e.g. a zero-mean Gaussian with variance $\sigma_n^2$ for each of the $N$ states. If observations are made at $T$ distinct time points the $N \times T$ matrices summarise the overall observed system as $\mathbf{Y} = \mathbf{X} + \mathbf{E}$. In order to obtain values for $\mathbf{X}$ the system of ODEs must be solved, so that in the case of an initial value problem $\mathbf{X}(\boldsymbol{\theta}, \mathbf{x}_0)$ denotes the solution of the system of equations at the specified time points for the parameters $\boldsymbol{\theta}$ and initial conditions $\mathbf{x}_0$. Figure 1(a) illustrates graphically the conditional dependencies of the overall statistical model and from this the posterior density follows by employing appropriate priors such that $p(\boldsymbol{\theta}, \mathbf{x}_0, \boldsymbol{\sigma}|\mathbf{Y}) \propto \pi(\boldsymbol{\theta})\pi(\mathbf{x}_0)\pi(\boldsymbol{\sigma}) \prod_n \mathcal{N}_{\mathbf{Y}_{n,\cdot}}(\mathbf{X}(\boldsymbol{\theta}, \mathbf{x}_0)_{n,\cdot}, \mathbf{I}\sigma_n^2)$. The desired marginal $p(\boldsymbol{\theta}|\mathbf{Y})$ can be obtained from this joint posterior[2].

Various sampling schemes can be devised to sample from the joint posterior. However, regardless of the sampling method, each proposal requires the specific solution of the system of differential equations which, as will be demonstrated in the experimental sections, is the main computational bottleneck in running an MCMC scheme for models based on differential equations. The computational complexity of numerically solving such a system cannot be easily quantified since it depends on many factors such as the type of model and its stiffness, which in turn depends on the specific parameter values used. A method to alleviate this bottleneck is the main contribution of this paper.

## 3 Auxiliary Gaussian Processes on State Variables

Let us assume independent[3] Gaussian process priors on the state variables such that $p(\mathbf{X}_{n,\cdot}|\boldsymbol{\varphi}_n) = \mathcal{N}(\mathbf{0}, \mathbf{C}_{\boldsymbol{\varphi}_n})$, where $\mathbf{C}_{\boldsymbol{\varphi}_n}$ denotes the matrix of covariance function values with hyperparameters $\boldsymbol{\varphi}_n$. With noise $\boldsymbol{\epsilon}_n \sim \mathcal{N}(\mathbf{0}, \sigma_n^2 \mathbf{I}_T)$, the state posterior, $p(\mathbf{X}_{n,\cdot}|\mathbf{Y}_{n,\cdot}, \sigma_n, \boldsymbol{\varphi}_n)$ follows as $\mathcal{N}(\boldsymbol{\mu}_n, \boldsymbol{\Sigma}_n)$ where $\boldsymbol{\mu}_n = \mathbf{C}_{\boldsymbol{\varphi}_n}(\mathbf{C}_{\boldsymbol{\varphi}_n} + \sigma_n^2 \mathbf{I})^{-1}\mathbf{Y}_{n,\cdot}$ and $\boldsymbol{\Sigma}_n = \sigma_n^2 \mathbf{C}_{\boldsymbol{\varphi}_n}(\mathbf{C}_{\boldsymbol{\varphi}_n} + \sigma_n^2 \mathbf{I})^{-1}$. Given priors $\pi(\sigma_n)$ and $\pi(\boldsymbol{\varphi}_n)$ the corresponding posterior is $p(\boldsymbol{\varphi}_n, \sigma_n|\mathbf{Y}_{n,\cdot}) \propto \pi(\sigma_n)\pi(\boldsymbol{\varphi}_n)\mathcal{N}_{\mathbf{Y}_{n,\cdot}}(\mathbf{0}, \sigma_n^2 \mathbf{I} + \mathbf{C}_{\boldsymbol{\varphi}_n})$ and from this we can obtain the joint posterior, $p(\mathbf{X}, \sigma_{n=1\cdots N}, \boldsymbol{\varphi}_{n=1\cdots N}|\mathbf{Y}, )$, over a non-parametric GP model of the state-variables. Note that a non-Gaussian noise model may alternatively be implemented using warped GPs [14]. The conditional distribution for the state-derivatives is

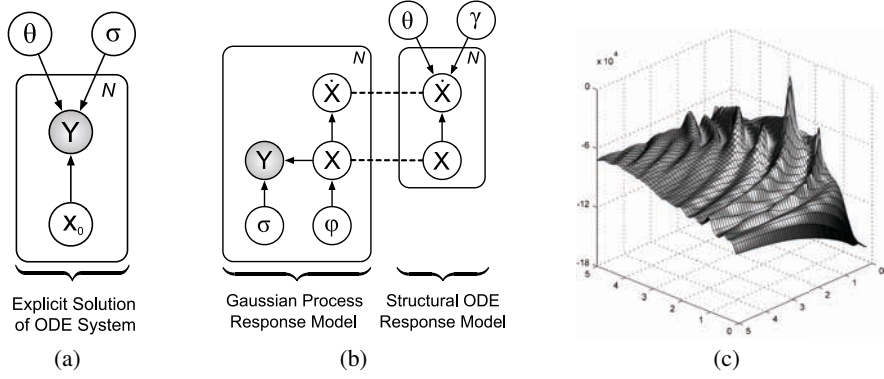

Figure 1: (a) Graphical model representing explicit solution of an ODE system, (b) Graphical model representing approach developed in this paper with dashed lines showing how the two models are combined in product form, (c) Likelihood surface for a simple oscillator model

$p(\dot{\mathbf{X}}_{n,\cdot}|\mathbf{X}_{n,\cdot}, \boldsymbol{\varphi}_n, \sigma_n) = \mathcal{N}(\mathbf{m}_n, \mathbf{K}_n)$, where the mean and covariance are given by

$$\mathbf{m}_n = {}'\mathbf{C}_{\boldsymbol{\varphi}_n}(\mathbf{C}_{\boldsymbol{\varphi}_n} + \sigma_n^2\mathbf{I})^{-1}\mathbf{X}_{n,\cdot}, \quad \text{and} \quad \mathbf{K}_n = \mathbf{C}_{\boldsymbol{\varphi}_n}'' - {}'\mathbf{C}_{\boldsymbol{\varphi}_n}(\mathbf{C}_{\boldsymbol{\varphi}_n} + \sigma_n^2\mathbf{I})^{-1}\mathbf{C}_{\boldsymbol{\varphi}_n}'$$

where $\mathbf{C}_{\boldsymbol{\varphi}_n}''$ denotes the auto-covariance for each state-derivative with $\mathbf{C}_{\boldsymbol{\varphi}_n}'$ and ${}'\mathbf{C}_{\boldsymbol{\varphi}_n}$ denoting the cross-covariances between the state and its derivative [13, 15]. The main advantage of using the Gaussian process model now becomes apparent. The GP specifies a jointly Gaussian distribution over the function and its derivatives ([13], pg.191). This allows us to evaluate a posterior over parameters $\boldsymbol{\theta}$ consistent with the differential equation based on the smoothed state and state derivative estimates, see Figure 1(b). Assuming Normal errors between the state-derivatives $\dot{\mathbf{X}}_{n,\cdot}$ and the functional, $\mathbf{f}_n(\mathbf{X}, \boldsymbol{\theta}, \mathbf{t})$ evaluated at the GP generated state-values, $\mathbf{X}$ corresponding to time points $\mathbf{t} = t_1 \cdots t_T$ then $p(\dot{\mathbf{X}}_{n,\cdot}|\mathbf{X}, \boldsymbol{\theta}, \gamma_n) = \mathcal{N}(\mathbf{f}_n(\mathbf{X}, \boldsymbol{\theta}, \mathbf{t}), \mathbf{I}\gamma_n)$ with $\gamma_n$ a state-specific error variance. Both statistical models $p(\dot{\mathbf{X}}_{n,\cdot}|\mathbf{X}_{n,\cdot}, \boldsymbol{\varphi}_n, \sigma_n)$ and $p(\dot{\mathbf{X}}_{n,\cdot}|\mathbf{X}, \boldsymbol{\theta}, \gamma_n)$ can be linked in the form of a Product of Experts [7] to define the overall density $p(\dot{\mathbf{X}}_{n,\cdot}|\mathbf{X}, \boldsymbol{\theta}, \gamma_n, \boldsymbol{\varphi}_n, \sigma_n) \propto \mathcal{N}(\mathbf{m}_n, \mathbf{K}_n)\mathcal{N}(\mathbf{f}_n(\mathbf{X}, \boldsymbol{\theta}, \mathbf{t}), \mathbf{I}\gamma_n)$ [see e.g. 20]. Introducing priors $\pi(\boldsymbol{\theta})$ and $\pi(\boldsymbol{\gamma}) = \prod_n \pi(\gamma_n)$

$$
\begin{aligned}
p(\boldsymbol{\theta}, \boldsymbol{\gamma}|\mathbf{X}, \boldsymbol{\varphi}, \boldsymbol{\sigma}) &= \int p(\dot{\mathbf{X}}, \boldsymbol{\theta}, \boldsymbol{\gamma}|\mathbf{X}, \boldsymbol{\varphi}, \boldsymbol{\sigma})d\dot{\mathbf{X}} \\
&\propto \pi(\boldsymbol{\theta})\pi(\boldsymbol{\gamma}) \prod_n \int \mathcal{N}(\mathbf{m}_n, \mathbf{K}_n)\mathcal{N}(\mathbf{f}_n(\mathbf{X}, \boldsymbol{\theta}, \mathbf{t}), \mathbf{I}\gamma_n)d\dot{\mathbf{X}}_{n,\cdot} \\
&\propto \frac{\pi(\boldsymbol{\theta})\pi(\boldsymbol{\gamma})}{\prod_n \mathcal{Z}(\gamma_n)} \exp\left\{-\frac{1}{2}\sum_n (\mathbf{f}_n - \mathbf{m}_n)^\mathsf{T}(\mathbf{K}_n + \mathbf{I}\gamma_n)^{-1}(\mathbf{f}_n - \mathbf{m}_n)\right\}
\end{aligned}
$$

where $\mathbf{f}_n \equiv \mathbf{f}_n(\mathbf{X}, \boldsymbol{\theta}, \mathbf{t})$, and $\mathcal{Z}(\gamma_n) = |2\pi(\mathbf{K}_n + \mathbf{I}\gamma_n)|^{\frac{1}{2}}$ is a normalizing constant. Since the gradients appear only linearly and their conditional distribution given $\mathbf{X}$ is Gaussian they can be marginalized exactly. In other words, given observations $\mathbf{Y}$, we can sample from the conditional distribution for $\mathbf{X}$ and marginalize the augmented derivative space. The differential equation need now never be explicitly solved, its implicit solution is integrated into the sampling scheme.

## 4 Sampling Schemes for Fully and Partially Observed Systems

The introduction of the auxiliary model and its associated variables has enabled us to recast the differential equation as another component of the inference process. The relationship between the auxiliary model and the physical process that we are modeling is shown in Figure 1(b), where the dotted lines represent a transfer of information between the models. This information transfer takes place through sampling candidate solutions for the system in the GP model. Inference is performed by combining these approximate solutions with the system dynamics from the differential equations. It now remains to define an overall sampling scheme for the structural parameters. For brevity, we

omit normalizing constants and assume that the system is defined in terms of ODEs. However, our scheme is easily extended for delay differential equations (DDEs) where now predictions at each time point $t$ and the associated delay $(t - \tau)$ are required — we present results for a DDE system in Section 5.2. We can now consider the complete sampling scheme by also inferring the hyperparameters and corresponding predictions of the state variables and derivatives using the GP framework described in Section 3. We can obtain samples $\boldsymbol{\theta}$ from the desired marginal posterior $p(\boldsymbol{\theta}|\mathbf{Y})^4$ by sampling from the joint posterior $p(\boldsymbol{\theta}, \boldsymbol{\gamma}, \mathbf{X}, \boldsymbol{\varphi}, \boldsymbol{\sigma}|\mathbf{Y})$ as follows

$$\boldsymbol{\varphi}_n, \sigma_n | \mathbf{Y}_{n,\cdot} \quad \sim \quad p(\boldsymbol{\varphi}_n, \sigma_n | \mathbf{Y}_{n,\cdot}) \propto \pi(\sigma_n)\pi(\boldsymbol{\varphi}_n)\mathcal{N}_{\mathbf{Y}_{n,\cdot}}(\mathbf{0}, \sigma_n^2\mathbf{I} + \mathbf{C}_{\boldsymbol{\varphi}_n}) \tag{1}$$

$$\mathbf{X}_{n,\cdot} | \mathbf{Y}_{n,\cdot}, \sigma_n, \boldsymbol{\varphi}_n \quad \sim \quad p(\mathbf{X}_{n,\cdot} | \mathbf{Y}_{n,\cdot}, \sigma_n, \boldsymbol{\varphi}_n) = \mathcal{N}_{\mathbf{X}_{n,\cdot}}(\boldsymbol{\mu}_n, \boldsymbol{\Sigma}_n) \tag{2}$$

$$\boldsymbol{\theta}, \boldsymbol{\gamma} | \mathbf{X}, \boldsymbol{\varphi}, \boldsymbol{\sigma} \quad \sim \quad p(\boldsymbol{\theta}, \boldsymbol{\gamma} | \mathbf{X}, \boldsymbol{\varphi}, \boldsymbol{\sigma}) \propto \pi(\boldsymbol{\theta})\pi(\boldsymbol{\gamma}) \exp\left\{ -\frac{1}{2}\sum_n \boldsymbol{\delta}_n^\mathsf{T}(\mathbf{K}_n + \mathbf{I}\gamma_n)^{-1}\boldsymbol{\delta}_n \right\} \tag{3}$$

where $\boldsymbol{\delta}_n \equiv \mathbf{f}_n - \mathbf{m}_n$. This requires two Metropolis sampling schemes; one for inferring the parameters of the GP, $\boldsymbol{\varphi}$ and $\boldsymbol{\sigma}$, and another for the parameters of the structural system, $\boldsymbol{\theta}$ and $\boldsymbol{\gamma}$. However, as a consequence of the system induced dynamics the corresponding likelihood surface defined by $p(\mathbf{Y}|\boldsymbol{\theta}, \mathbf{x}_0, \boldsymbol{\sigma})$ can present formidable challenges to standard sampling methods. As an example Figure 1(c) illustrates the induced likelihood surface of a simple dynamic oscillator similar to that presented in the experimental section. Recent advances in MCMC methodology suggest solutions to this problem in the form of population-based MCMC methods [8], which we therefore implement to sample the structural parameters of our model. Population MCMC enables samples to be drawn from a target density $p(\boldsymbol{\theta})$ by defining a product of annealed densities indexed by a temperature parameter $\boldsymbol{\beta}$, such that $p(\boldsymbol{\theta}|\boldsymbol{\beta}) = \prod_i p(\boldsymbol{\theta}|\beta_i)$ and the desired target density $p(\boldsymbol{\theta})$ is defined for one value of $\beta_i$. It is convenient to fix a geometric path between the prior and posterior, which we do in our implementation, although other sequences are possible [3]. A time homogeneous Markov transition kernel which has $p(\boldsymbol{\theta})$ as its stationary distribution can then be constructed from both local Metropolis proposal moves and global temperature switching moves between the tempered chains of the population [8], allowing freer movement within the parameter space.

The computational scaling for each component of the sampler is now considered. Sampling of the GP covariance function parameters by a Metropolis step requires computation of a matrix determinant and its inverse, so for all $N$ states in the system a dominant scaling of $\mathcal{O}(NT^3)$ will be obtained. This poses little problem for many applications in systems biology since $T$ is often fairly small ($T \approx 10$ to $100$). For larger values of $T$, sparse approximations can offer much improved computational scaling of order $\mathcal{O}(NM^2T)$, where $M$ is the number of time points selected [9]. Sampling from a multivariate Normal whose covariance matrix and corresponding decompositions have already been computed therefore incurs no dominating additional computational overhead. The final Metropolis step (Equation 3) requires each of the $\mathbf{K}_n$ matrices to be constructed and the associated determinants and inverses computed thus incurring a total $\mathcal{O}(NT^3)$ scaling per sample.

An approximate scheme can be constructed by first obtaining the *maximum a posteriori* values for the GP hyperparameters and posterior mean state values, $\hat{\boldsymbol{\varphi}}$, $\hat{\boldsymbol{\sigma}}$, $\hat{\mathbf{X}}_n$, and then employing these in Equation 3. This will provide samples from $p(\boldsymbol{\theta}, \boldsymbol{\gamma}|\hat{\mathbf{X}}, \hat{\boldsymbol{\varphi}}, \hat{\sigma}, \mathbf{Y})$ which may be a useful surrogate for the full joint posterior incurring lower computational cost as all matrix operations will have been pre-computed, as will be demonstrated later in the paper.

We can also construct a sampling scheme for the important special case where some states are unobserved. We partition $\mathbf{X}$ into $\mathbf{X}_o$, and $\mathbf{X}_u$. Let $o$ index the observed states, then we may infer all the unknown variables as follows

$$p(\boldsymbol{\theta}, \boldsymbol{\gamma}, \mathbf{X}_u | \mathbf{X}_o, \boldsymbol{\varphi}, \boldsymbol{\sigma}) \quad \propto \quad \pi(\boldsymbol{\theta})\pi(\boldsymbol{\gamma})\pi(\mathbf{X}_u) \exp\left\{ -\frac{1}{2}\sum_{n \in o}(\boldsymbol{\delta}_n^{o,u})^\mathsf{T}(\mathbf{K}_n + \mathbf{I}\gamma_n)^{-1}(\boldsymbol{\delta}_n^{o,u}) \right\}$$

where $\boldsymbol{\delta}_n^{o,u} \equiv \mathbf{f}_n(\mathbf{X}_o, \mathbf{X}_u, \boldsymbol{\theta}, \mathbf{t}) - \mathbf{m}_n$ and $\pi(\mathbf{X}_u)$ is an appropriately chosen prior. The values of unobserved species are obtained by propagating their sampled initial values using the corresponding discrete versions of the differential equations and the smoothed estimates of observed species. The p53 transcriptional network example we include requires inference over unobserved protein species, see Section 5.3.

# 5 Experimental Examples

We now demonstrate our GP-based method using a standard squared exponential covariance function on a variety of examples involving both ordinary and delay differential equations, and compare the accuracy and speed with other state-of-the-art methods.

## 5.1 Example 1 - Nonlinear Ordinary Differential Equations

We first consider the FitzHugh-Nagumo model [11] which was originally developed to model the behaviour of spike potentials in the giant axon of squid neurons and is defined as $\dot{V} = c\left(V - V^3/3 + R\right)$, $\dot{R} = -\left(V - a + bR\right)/c$. Although consisting of only 2 equations and 3 parameters, this dynamical system exhibits a highly nonlinear likelihood surface [11], which is induced by the sharp changes in the properties of the limit cycle as the values of the parameters vary. Such a feature is common to many nonlinear systems and so this model provides an excellent test for our GP-based parameter inference method.

Data is generated from the model, with parameters $a = 0.2$, $b = 0.2$, $c = 3$, at $\{40, 80, 120\}$ time points with additive Gaussian noise, $N(0, v)$ for $v = 0.1 \times \sigma_n$, where $\sigma_n$ is the standard deviation for the $n$th species. The parameters were then inferred from these data sets using the full Bayesian sampling scheme and the approximate sampling scheme (Section 4), both employing population MCMC. Additionally, we inferred the parameters using 2 alternative methods, the profiled estimation method of Ramsay et al. [11] and a Population MCMC based sampling scheme, in which the ODEs were solved explicitly (Section 2), to complete the comparative study. All the algorithms were coded in Matlab, and the population MCMC algorithms were run with 30 temperatures, and used a suitably diffuse $\Gamma(2, 1)$ prior distribution for all parameters, forming the base distribution for the sampler. Two of these population MCMC samplers were run in parallel and the $\hat{R}$ statistic [5] was used to monitor convergence of all chains at all temperatures. The required numerical approximations to the ODE were calculated using the Sundials ODE solver, which has been demonstrated to be considerably (up to 100 times) faster than the standard ODE45/ODE15s solvers commonly used in Matlab. In our experiments the chains generally converged after around 5000 iterations, and 2000 samples were then drawn to form the posterior distributions. Ramsay's method [11] was implemented using the Matlab code which accompanies their paper. The optimal algorithm settings were used, tuned for the FitzHugh-Nagumo model (see [11] for details) which they also investigated. Each experiment was repeated 100 times, and Table 1 shows summary statistics for each of the inferred parameters. All of the three sampling methods based on population MCMC produced low variance samples from posteriors positioned close to the true parameters values. Most noticeable from the results in Figure 2 is the dramatic speed advantage the GP based methods have over the more direct approach, whereby the differential equations are solved explicitly; the GP methods introduced in this paper offer up to a 10-fold increase in speed, even for this relatively simple system of ODEs. We found the performance of the profiled estimation method [11] to be very sensitive to the initial parameter values. In practice parameter values are unknown, indeed little may be known even about the range of possible values they may take. Thus it seems sensible to choose initial values from a wide prior distribution so as to explore as many regions of parameter space as possible. Employing

| FitzHugh-Nagumo ODE Model | | | | |
|---|---|---|---|---|
| Samples | Method | $a$ | $b$ | $c$ |
| 40 | GP MAP | $0.1930 \pm 0.0242$ | $0.2070 \pm 0.0453$ | $2.9737 \pm 0.0802$ |
| | GP Fully Bayesian | $0.1983 \pm 0.0231$ | $0.2097 \pm 0.0481$ | $3.0133 \pm 0.0632$ |
| | Explicit ODE | $0.2015 \pm 0.0107$ | $0.2106 \pm 0.0385$ | $3.0153 \pm 0.0247$ |
| 80 | GP MAP | $0.1950 \pm 0.0206$ | $0.2114 \pm 0.0386$ | $2.9801 \pm 0.0689$ |
| | GP Fully Bayesian | $0.2068 \pm 0.0194$ | $0.1947 \pm 0.0413$ | $3.0139 \pm 0.0585$ |
| | Explicit ODE | $0.2029 \pm 0.0121$ | $0.1837 \pm 0.0304$ | $3.0099 \pm 0.0158$ |
| 120 | GP MAP | $0.1918 \pm 0.0145$ | $0.2088 \pm 0.0317$ | $3.0137 \pm 0.0489$ |
| | GP Fully Bayesian | $0.1971 \pm 0.0162$ | $0.2081 \pm 0.0330$ | $3.0069 \pm 0.0593$ |
| | Explicit ODE | $0.2071 \pm 0.0112$ | $0.2123 \pm 0.0286$ | $3.0112 \pm 0.0139$ |

Table 1: Summary statistics for each of the inferred parameters of the FitzHugh-Nagumo model. Each experiment was repeated 100 times and the mean parameter values are shown. We observe that all three population-based MCMC methods converge close to the true parameter values, $a = 0.2$, $b = 0.2$ and $c = 3$.

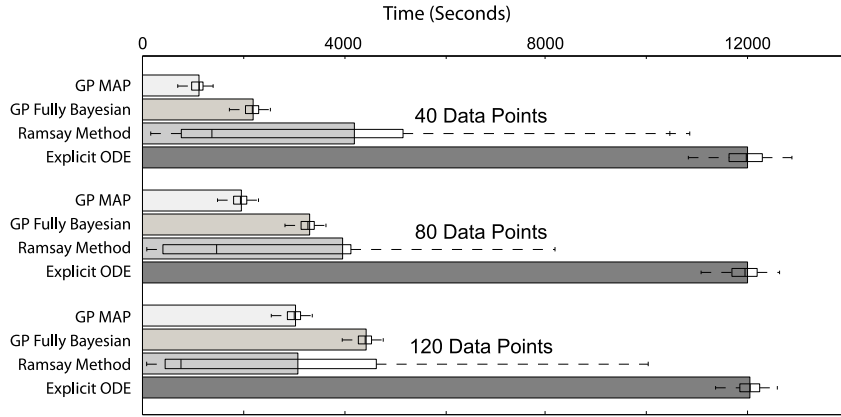

Figure 2: Summary statistics of the overall time taken for the algorithms to run to completion. Solid bars show mean time for 100 runs; superimposed boxplots display median results with upper and lower quartiles.

profiled estimation using initial parameter values drawn from a wide gamma prior, however, yielded highly biased results, with the algorithm often converging to local maxima far from the true parameter values. The parameter estimates become more biased as the variance of the prior is increased, i.e. as the starting points move further from the true parameter values. E.g. consider parameter $a$; for 40 data points, for initial values $a, b, c \sim \mathcal{N}(\{0.2, 0.2, 3\}, 0.2)$, the range of estimated values for $\hat{a}$ was [Min, Median, Max] $= [0.173, 0.203, 0.235]$. For initial values $a, b, c \sim \Gamma(1, 0.5)$, the $\hat{a}$ had a range [Min, Median, Max] $= [-0.329, 0.205, 9.3 \times 10^9]$ and for a wider prior $a, b, c \sim \Gamma(2, 1)$, then $\hat{a}$ had range [Min, Median, Max] $= [-1.4 \times 10^{10}, 0.195, 2.2 \times 10^9]$. Lack of robustness therefore seems to be a significant problem with this profiled estimation method. The speed of the profiled estimation method was also extremely variable, and this was observed to be very dependent on the initial parameter values e.g. for initial values $a, b, c \sim \mathcal{N}(\{0.2, 0.2, 3\}, 0.2)$, the times recorded were [Min, Mean, Max] $= [193, 308, 475]$. Using a different prior for initial values such that $a, b, c \sim \Gamma(1, 0.5)$, the times were [Min, Mean, Max] $= [200, 913, 3265]$ and similarly for a wider prior $a, b, c \sim \Gamma(2, 1)$, [Min, Mean, Max] $= [132, 4171, 37411]$. Experiments performed with noise $v = \{0.05, 0.2\} \times \sigma_n$ produced similar and consistent results, however they are omitted due to lack of space.

## 5.2 Example 2 - Nonlinear Delay Differential Equations

This example model describes the oscillatory behaviour of the concentration of mRNA and its corresponding protein level in a genetic regulatory network, introduced by Monk [10]. The translocation of mRNA from the nucleus to the cytosol is explicitly described by a delay differential equation.

$$\frac{d\mu}{dt} = \frac{1}{1 + (p(t-\tau)/p_0)^n} - \mu_m \mu \qquad \frac{dp}{dt} = \mu - \mu_p p$$

where $\mu_m$ and $\mu_p$ are decay rates, $p_0$ is the repression threshold, $n$ is a Hill coefficient and $\tau$ is the time delay. The application of our method to DDEs is of particular interest since numerical solutions to DDEs are generally much more computationally expensive to obtain than ODEs. Thus inference of such models using MCMC methods and explicitly solving the system at each iteration becomes less feasible as the complexity of the system of DDEs increases.

We consider data generated from the above model, with parameters $\mu_m = 0.03$, $\mu_p = 0.03$, $p_0 = 100$, $\tau = 25$, at $\{40, 80, 120\}$ time points with added random noise drawn from a Gaussian distribution, $N(0, v)$ for $v = 0.1 \times \sigma_n$, where $\sigma_n$ is the standard deviation of the time-series data for the $n$th species. The parameters were then inferred from these data sets using our GP-based population MCMC methods. Figure 3 shows a time comparison for 10 iterations of the GP sampling algorithms and compares it to explicitly solving the DDEs using the Matlab solver DDE23 (which is generally faster than the Sundials solver for DDEs). The GP methods are around 400 times faster for 40 data points. Using the GP methods, samples from the full posterior can be obtained in less than an hour. Solving the DDEs explicitly, the population MCMC algorithm would take in excess of two weeks computation time, assuming the chains take a similar number of iterations to converge.

| Monk DDE Model | | | | | |
|---|---|---|---|---|---|
| Samples | Method | $\mu_m$ | $\mu_p \times 10^{-3}$ | $p_0 \times 10^{-3}$ | $\tau$ |
| 40 | GP MAP | $100.21 \pm 2.08$ | $29.7 \pm 1.6$ | $30.1 \pm 0.3$ | $25.65 \pm 1.04$ |
| | GP Full Bayes | $99.75 \pm 1.50$ | $29.8 \pm 1.2$ | $30.1 \pm 0.2$ | $25.33 \pm 0.85$ |
| 80 | GP MAP | $99.48 \pm 1.29$ | $29.5 \pm 0.9$ | $30.1 \pm 0.1$ | $24.81 \pm 0.59$ |
| | GP Full Bayes | $100.26 \pm 1.03$ | $30.1 \pm 0.6$ | $30.1 \pm 0.1$ | $24.87 \pm 0.44$ |
| 120 | GP MAP | $99.91 \pm 1.02$ | $30.0 \pm 0.5$ | $30.0 \pm 0.1$ | $24.97 \pm 0.38$ |
| | GP Full Bayes | $100.23 \pm 0.92$ | $30.0 \pm 0.4$ | $30.0 \pm 0.1$ | $25.03 \pm 0.25$ |

Table 2: Summary statistics for each of the inferred parameters of the Monk model. Each experiment was repeated 100 times and we observe that both GP population-based MCMC methods converge close to the true parameter values, $\mu_m = 100$, $\mu_p = 30 \times 10^{-3}$ and $p_0 = 30 \times 10^{-3}$. The time-delay parameter, $\tau = 25$, is also successfully inferred.

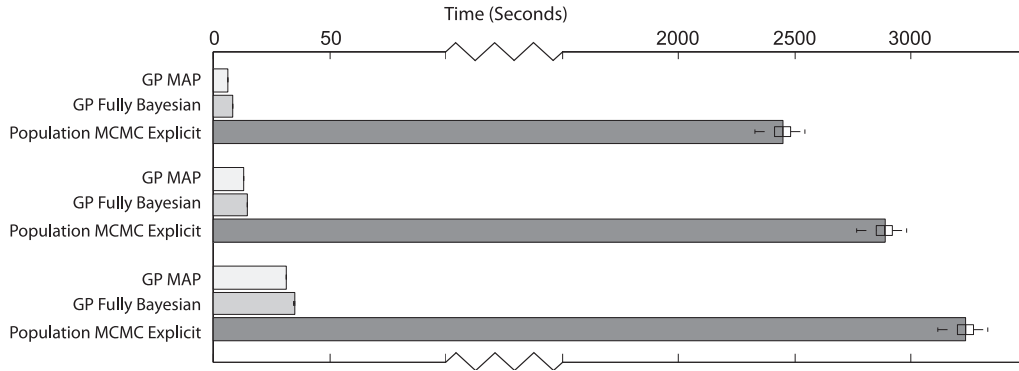

Figure 3: Summary statistics of the time taken for the algorithms to complete 10 iterations using DDE model.

## 5.3   Example 3 - The p53 Gene Regulatory Network with Unobserved Species

Our third example considers a linear and a nonlinear model describing the regulation of 5 target genes by the tumour repressor transcription factor protein p53. We consider the following differential equations which relate the expression level $x_j(t)$ of the $j$th gene at time $t$ to the concentration of the transcription factor protein $f(t)$ which regulates it, $\dot{x}_j = B_j + S_j g(f(t)) - D_j x_j(t)$, where $B_j$ is the basal rate of gene $j$, $S_j$ is the sensitivity of gene $j$ to the transcription factor and $D_j$ is the decay rate of the mRNA. Letting $g(f(t)) = f(t)$ gives us the linear model originally investigated in [1], and letting $g(f(t)) = \exp(f(t))$ gives us the nonlinear model investigated in [4]. The transcription factor $f(t)$ is unobserved and must be inferred along with the other structural parameters $B_j$, $S_j$ and $D_j$ using the sampling scheme detailed in Section 4.1. In this experiment, priors on the unobserved species used were $f(t) \sim \Gamma(2, 1)$ with a log-Normal proposal. We test our method using the

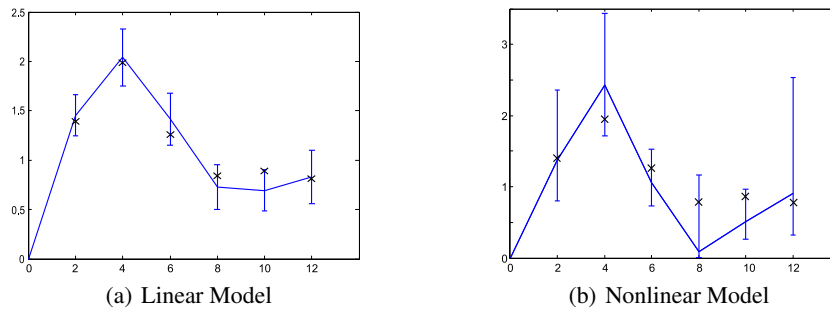

(a) Linear Model          (b) Nonlinear Model

Figure 4: The predicted output of the p53 gene using data from Barenco et al. [1] and the accelerated GP inference method for (a) the linear model and (b) the nonlinear response model. Note that the asymmetric error bars in (b) are due to $exp(y)$ being plotted, as opposed to just $y$ in (a). Our results are compared to the results obtained by Barenco et al. [1] (shown as crosses) and are comparable to those obtained by Lawrence et al. [4].

leukemia data set studied in [1], which comprises 3 measurements at each of 7 time points for each of the 5 genes. Figure 4 shows the inferred missing species and the results are in good accordance with recent biological studies. For this example, our GP sampling algorithms ran to completion in under an hour on a 2.2GHz Centrino laptop, with no difference in speed between using the linear and nonlinear models; indeed the equations describing this biological system could be made more complex with little additional computational cost.

## 6  Conclusions

Explicit solution of differential equations is a major bottleneck for the application of inferential methodology in a number of application areas, e.g. systems biology, nonlinear dynamic systems. We have addressed this problem and placed it within a Bayesian framework which tackles the main shortcomings of previous solutions to the problem of system identification for nonlinear differential equations. Our methodology allows the possibility of model comparison via the use of Bayes factors, which may be straightforwardly calculated from the samples obtained from the population MCMC algorithm. Possible extensions to this method include more efficient sampling exploiting control variable methods [17], embedding characteristics of a dynamical system in the design of covariance functions and application of our method to models involving partial differential equations.

**Acknowledgments**

Ben Calderhead is supported by Microsoft Research through its European PhD Scholarship Programme. Mark Girolami is supported by an EPSRC Advanced Research Fellowship EP/EO52029 and BBSRC Research Grant BB/G006997/1.

**References**

[1] Barenco, M., Tomescu, D., Brewer, D., Callard, D., Stark, J. and Hubank, M. (2006) Ranked prediction of p53 targets using hidden variable dynamic modeling, *Genome Biology*, **7** (3):R25.

[2] Doucet, A., de Freitas, N. and Gordon, N., (2001) Sequential Monte Carlo Methods in Practice, *Springer*.

[3] Friel, N. and Pettitt, A. N. (2008) Marginal Likelihood Estimation via Power Posteriors. *Journal of the Royal Statistical Society: Series B*, **70** (3), 589-607.

[4] Gao, P., Honkela, A., Rattray, M. and Lawrence, N.D. (2008) Gaussian Process Modelling of Latent Chemical Species: Applications to Inferring Transcription Factor Activities, *Bioinformatics*, **24**, i70-i75.

[5] Gelman, A., Carlin, J.B., Stern, H.S. and Rubin, D.B. (2004) Bayesian Data Analysis, *Chapman & Hall*.

[6] Graepel, T., (2003) Solving noisy linear operator equations by Gaussian processes: application to ordinary and partial differential equations, *Proc. ICML 2003*.

[7] Mayraz, G. and Hinton, G. (2001) Recognizing Hand-Written Digits Using Hierarchical Products of Experts, *Proc. NIPS 13*.

[8] Jasra, A., Stephens, D.A. and Holmes, C.C., (2007) On population-based simulation for static inference, *Statistics and Computing*, **17**, 263-279.

[9] Lawrence, N.D., Seeger, M. and Herbrich, R. (2003) Fast sparse Gaussian process methods: the informative vector machine, *Proc. NIPS 15*.

[10] Monk, N. (2003) Oscillatory Expression of Hes1, p53, and NF-kB Driven by Transcriptional Time Delays. *Current Biology*, **13** (16), 1409-1413.

[11] Ramsay, J., Hooker, G., Campbell, D. and Cao, J. (2007) Parameter Estimation for Differential Equations: A Generalized Smoothing Approach. *Journal of the Royal Statistical Society: Series B*, **69** (5), 741-796.

[12] Rasmussen, C, E., (2003) Gaussian processes to speed up hybrid Monte Carlo for expensive Bayesian integrals, *Bayesian Statistics*, **7**, 651-659.

[13] Rasmussen, C.E. and Williams, C.K.I. (2006) Gaussian Processes for Machine Learning, *The MIT Press*.

[14] Snelson, E., Rasmussen, C.E. and Ghahramani, Z. (2004), Warped Gaussian processes, *Proc. NIPS 16*.

[15] Solak, E., Murray-Smith, R., Leithead, W.E., Leith, D.J. and Rasmussen, C.E. (2003) Derivative observations in Gaussian Process models of dynamic systems, *Proc. NIPS 15*.

[16] Tarantola, A. (2005) Inverse Problem Theory and Methods for Model Parameter Estimation, *SIAM*.

[17] Titsias, M. and Lawrence, N. (2008) Efficient Sampling for Gaussian Process Inference using Control Variables, *Proc. NIPS 22*.

[18] Varah, J.M. (1982) A spline least squares method for numerical parameter estimation in differential equations. *SIAM J. Scient. Comput.*, **3**, 28-46.

[19] Vyshemirsky, V. and and Girolami, M., (2008), Bayesian ranking of biochemical system models *Bioinformatics* **24**, 833-839.

[20] Williams, C.K.I., Agakov, F.V., Felderof, S.N. (2002), Products of Gaussians, *Proc. NIPS 14*.

## Footnotes

[1]The methodology in this paper can also be straightforwardly extended to partial differential equations.

[2]This distribution is implicitly conditioned on the numerical solver and associated error tolerances.

[3]The dependencies between state variables can be modeled by defining the overall state vector as $\mathbf{x} = \mathsf{vec}(\mathbf{X})$ and using a GP prior of the form $\mathbf{x} \sim \mathcal{N}(\mathbf{0}, \boldsymbol{\Sigma} \otimes \mathbf{C})$ where $\otimes$ denotes the Kronecker matrix product and $\boldsymbol{\Sigma}$ is an $N \times N$ positive semi-definite matrix specifying inter-state similarities with $\mathbf{C}$, the $T \times T$ matrix defining intra-state similarities [13].

[4]Note that this is implicitly conditioned on the class of covariance function chosen.
